# Multilinear Subspace Regression: An Orthogonal Tensor Decomposition Approach

**Qibin Zhao** [1], **Cesar F. Caiafa** [2], **Danilo P. Mandic** [3], **Liqing Zhang**[4], **Tonio Ball** [5], **Andreas Schulze-Bonhage**[5], **and Andrzej Cichocki**[1]

[1]Brain Science Institute, RIKEN, Japan
[2]Instituto Argentino de Radioastronomía (IAR), CONICET, Argentina
[3]Dept. of Electrical & Electronic Engineering, Imperial College, UK
[4]Dept. of Computer Science & Engineering, Shanghai Jiao Tong University, China
[5]BCCN, Albert-Ludwigs-University, Germany
`qbzhao@brain.riken.jp`

## Abstract

A multilinear subspace regression model based on so called latent variable decomposition is introduced. Unlike standard regression methods which typically employ matrix (2D) data representations followed by vector subspace transformations, the proposed approach uses tensor subspace transformations to model common latent variables across both the independent and dependent data. The proposed approach aims to maximize the correlation between the so derived latent variables and is shown to be suitable for the prediction of multidimensional dependent data from multidimensional independent data, where for the estimation of the latent variables we introduce an algorithm based on Multilinear Singular Value Decomposition (MSVD) on a specially defined cross-covariance tensor. It is next shown that in this way we are also able to unify the existing Partial Least Squares (PLS) and N-way PLS regression algorithms within the same framework. Simulations on benchmark synthetic data confirm the advantages of the proposed approach, in terms of its predictive ability and robustness, especially for small sample sizes. The potential of the proposed technique is further illustrated on a real world task of the decoding of human intracranial electrocorticogram (ECoG) from a simultaneously recorded scalp electroencephalograph (EEG).

## 1   Introduction

The recent progress in sensor technology has made possible a plethora of novel applications, which typically require increasingly large amount of multidimensional data, such as large-scale images, 3D video sequences, and neuroimaging data. To match the data dimensionality, tensors (also called multiway arrays) have been proven to be a natural and efficient representation for such massive data. In particular, tensor subspace learning methods have been shown to outperform their corresponding vector subspace methods, especially for small sample size problems [1, 2]; these methods include multilinear PCA [3], multilinear LDA [4, 5], multiway covariates regression [6] and tensor subspace analysis [7]. These desirable properties have made tensor decomposition becoming a promising tool in exploratory data analysis [8, 9, 10, 11].

The Partial Least Squares (PLS) is a well-established estimation, regression and classification framework that aims to predict a set of dependent variables (responses) $\mathbf{Y}$ from a large set of independent variables (predictors) $\mathbf{X}$, and has been proven to be particularly useful for highly collinear data [12]. Its optimization objective is to maximize pairwise covariance of a set of latent variables (also called latent vectors, score vectors) by projecting both $\mathbf{X}$ and $\mathbf{Y}$ onto a new subspace. A popular way

to estimate the model parameters is the Non-linear Iterative Partial Least Squares (NIPALS) [13], an iterative procedure similar to the power method; for an overview of PLS and its applications in multivariate regression analysis see [14, 15, 16]. As an extension of PLS to multiway data, the $N$-way PLS (NPLS) decomposes the independent and dependent data into rank-one tensors, subject to maximum pairwise covariance of the latent vectors [17]. The widely reported sensitivity to noise of PLS is attributed to redundant (irrelevant) latent variables, whose selection remains an open problem. The number of latent variables also dependents on the rank of independent data, resulting in overfitting when the number of observations is smaller than the number of latent variables. Although the standard PLS can also handle an $N$-way tensor dataset differently, e.g. applied on a mode-1 matricization of $\underline{\mathbf{X}}$ and $\underline{\mathbf{Y}}$, this would make it difficult to interpret the loadings as the physical meaning would be lost due to the unfolding.

To alleviate these issues, in this study, a new tensor subspace regression model, called the Higer-Order Partial Least Squares (HOPLS), is proposed to predict an $M$th-order tensor $\underline{\mathbf{Y}}$ from an $N$th-order tensor $\underline{\mathbf{X}}$. It considers each data sample as a higher order tensor represented as a linear combination of tensor subspace bases. This way, the dimensionality of parameters estimated by HOPLS is much smaller than the dimensionality of parameters estimated by PLS, thus making HOPLS particularly suited for small sample sizes. In addition, the latent variables and tensor subspace can be optimized to ensure a maximum correlation between the latent variables of $\underline{\mathbf{X}}$ and $\underline{\mathbf{Y}}$ with a constraint imposed to ensure a special structure of the core tensor. This is achieved by a simultaneous stepwise rank-$(1, L_2, \ldots, L_N)$ decompositions of $\underline{\mathbf{X}}$ and rank-$(1, K_2, \ldots, K_M)$ decomposition of $\underline{\mathbf{Y}}$ [18], using multiway singular value decomposition (MSVD) [19].

## 2 Preliminaries

### 2.1 Notation and definitions

We denote $N$th-order tensors (*multi-way arrays*) by underlined boldface capital letters, matrices (*two-way arrays)* by boldface capital letters, and vectors by boldface lower-case letters, e.g., $\underline{\mathbf{X}}$, $\mathbf{P}$ and $\mathbf{t}$ are examples of a tensor, a matrix and a vector, respectively.

The $i$th entry of a vector $\mathbf{x}$ is denoted by $x_i$, element $(i, j)$ of a matrix $\mathbf{X}$ by $x_{ij}$, and element $(i_1, i_2, \ldots, i_N)$ of an $N$th-order tensor $\underline{\mathbf{X}} \in \mathbb{R}^{I_1 \times I_2 \times \cdots \times I_N}$ by $x_{i_1 i_2 \ldots i_N}$ or $(\underline{\mathbf{X}})_{i_1 i_2 \ldots i_N}$. Indices typically range from 1 to their capital version, e.g., $i_N = 1, \ldots, I_N$. The $n$th matrix in a sequence is denoted by a superscript in parentheses, e.g., $\mathbf{X}^{(n)}$. The $n$th-mode matricization of a tensor $\underline{\mathbf{X}}$ is denoted by $\mathbf{X}_{(n)}$.

The *n-mode product* of a tensor $\underline{\mathbf{X}} \in \mathbb{R}^{I_1 \times \cdots \times I_n \times \cdots \times I_N}$ and matrix $\mathbf{A} \in \mathbb{R}^{J_n \times I_n}$ is denoted by $\underline{\mathbf{Y}} = \underline{\mathbf{X}} \times_n \mathbf{A} \in \mathbb{R}^{I_1 \times \cdots \times I_{n-1} \times J_n \times I_{n+1} \times \cdots \times I_N}$ and is defined as:

$$y_{i_1 i_2 \ldots i_{n-1} j_n i_{n+1} \ldots i_N} = \sum_{i_n} x_{i_1 i_2 \ldots i_n \ldots i_N} a_{j_n i_n}. \tag{1}$$

The *n-mode cross-covariance* between an $N$th-order tensor $\underline{\mathbf{X}} \in \mathbb{R}^{I_1 \times \cdots \times I_n \times \cdots \times I_N}$ and an $M$th-order tensor $\underline{\mathbf{Y}} \in \mathbb{R}^{J_1 \times \cdots \times J_n \times \cdots \times J_M}$ with the same size $I_n = J_n$ on the $n$th-mode, denoted by $\mathrm{COV}_{\{n;n\}}(\underline{\mathbf{X}}, \underline{\mathbf{Y}}) \in \mathbb{R}^{I_1 \times \cdots \times I_{n-1} \times I_{n+1} \times \cdots \times I_N \times J_1 \times \cdots \times J_{n-1} \times J_{n+1} \times \cdots \times J_M}$, is defined as

$$\underline{\mathbf{C}} = \mathrm{COV}_{\{n;n\}}(\underline{\mathbf{X}}, \underline{\mathbf{Y}}) = < \underline{\mathbf{X}}, \underline{\mathbf{Y}} >_{\{n;n\}}, \tag{2}$$

where the symbol $< \bullet, \bullet >_{\{n;n\}}$ represents a multiplication between two tensors, and is defined as

$$c_{i_1, \ldots, i_{n-1}, i_{n+1} \ldots i_N, j_1, \ldots, j_{n-1} j_{n+1} \ldots j_M} = \sum_{i_n=1}^{I_n} x_{i_1, \ldots, i_n, \ldots, i_N} y_{j_1, \ldots, i_n, \ldots, j_M}. \tag{3}$$

## 2.2  Partial Least Squares

The objective of the PLS method is to find a set of latent vectors that explains as much as possible the covariance between $\mathbf{X}$ and $\mathbf{Y}$, which can be achieved by performing the following decomposition

$$\mathbf{X} = \mathbf{TP}^T + \mathbf{E} = \sum_{r=1}^{R} \mathbf{t}_r \mathbf{p}_r^T + \mathbf{E},$$

$$\mathbf{Y} = \mathbf{UC}^T + \mathbf{F} = \sum_{r=1}^{R} \mathbf{u}_r \mathbf{c}_r^T + \mathbf{F}, \tag{4}$$

where $\mathbf{T} = [\mathbf{t}_1, \mathbf{t}_2, \ldots, \mathbf{t}_R] \in \mathbb{R}^{I \times R}$ is a matrix of $R$ extracted orthogonal latent variables from $\mathbf{X}$, that is, $\mathbf{T}^T \mathbf{T} = \mathbf{I}$, and $\mathbf{U} = [\mathbf{u}_1, \mathbf{u}_2, \ldots, \mathbf{u}_R] \in \mathbb{R}^{I \times R}$ are latent variables from $\mathbf{Y}$ that have maximum covariance with $\mathbf{T}$ column-wise. The matrices $\mathbf{P}$ and $\mathbf{C}$ represent loadings (vector subspace bases) and $\mathbf{E}$ and $\mathbf{F}$ are residuals. A useful property is that the relation between $\mathbf{T}$ and $\mathbf{U}$ can be approximated linearly by

$$\mathbf{U} \approx \mathbf{TD}, \tag{5}$$

where $\mathbf{D}$ is an $(R \times R)$ diagonal matrix, and scalars $d_{rr} = \mathbf{u}_r^T \mathbf{t}_r / \mathbf{t}_r^T \mathbf{t}_r$ play the role of regression coefficients.

## 3  Higher-order PLS (HOPLS)

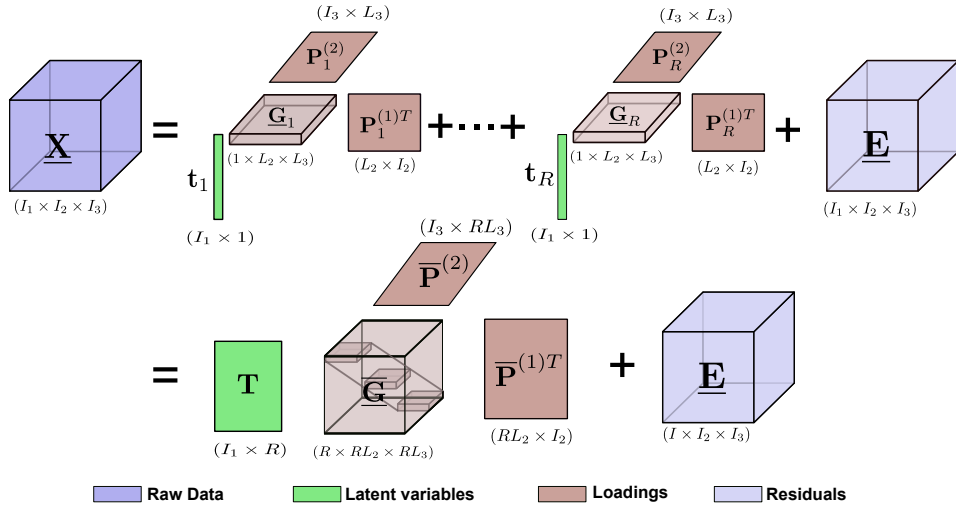

Figure 1: Schematic diagram of the HOPLS model: decomposing $\underline{\mathbf{X}}$ as a sum of rank-$(1, L_2, L_3)$ tensors. Decomposition for $\underline{\mathbf{Y}}$ follows a similar principle.

For an $N$th-order independent tensor $\underline{\mathbf{X}} \in \mathbb{R}^{I_1 \times \cdots \times I_N}$ and an $M$th-order dependent tensor $\underline{\mathbf{Y}} \in \mathbb{R}^{J_1 \times \cdots \times J_M}$, having the same size on the first mode[1], i.e., $I_1 = J_1$, similar to PLS, our objective is to find the optimal subspace approximation of $\underline{\mathbf{X}}$ and $\underline{\mathbf{Y}}$, in which the latent vectors of independent and dependent variables have maximum pairwise covariance.

### 3.1  Proposed model

The new tensor subspace represented by the Tucker model can be obtained by approximating $\underline{\mathbf{X}}$ with a sum of rank-$(1, L_2, \ldots, L_N)$ decompositions (see Fig.1), while dependent data $\underline{\mathbf{Y}}$ are approximated by a sum of rank-$(1, K_2, \ldots, K_M)$ decompositions. From the relation between the

latent vectors in (5), upon replacing $\mathbf{U}$ by $\mathbf{TD}$ and integrating $\mathbf{D}$ into the core tensor, the operation of HOPLS can be expressed as

$$\underline{\mathbf{X}} = \sum_{r=1}^{R} \underline{\mathbf{G}}_r \times_1 \mathbf{t}_r \times_2 \mathbf{P}_r^{(1)} \times_3 \cdots \times_N \mathbf{P}_r^{(N-1)} + \underline{\mathbf{E}},$$

$$\underline{\mathbf{Y}} = \sum_{r=1}^{R} \underline{\mathbf{D}}_r \times_1 \mathbf{t}_r \times_2 \mathbf{Q}_r^{(1)} \times_3 \cdots \times_M \mathbf{Q}_r^{(M-1)} + \underline{\mathbf{F}}, \tag{6}$$

where $R$ is the number of latent vectors, $\mathbf{t}_r \in \mathbb{R}^{I_1}$ is the $r$-th latent vector, $\left\{ \mathbf{P}_r^{(n)} \right\}_{n=1}^{N-1} \in \mathbb{R}^{I_{n+1} \times L_{n+1}} (I_{n+1} \geqslant L_{n+1})$ and $\left\{ \mathbf{Q}_r^{(m)} \right\}_{m=1}^{M-1} \in \mathbb{R}^{J_{n+1} \times K_{n+1}} (J_{n+1} \geqslant K_{n+1})$ are loading matrices corresponding to the latent vector $\mathbf{t}_r$ on mode-$n$ and mode-$m$ respectively, and $\underline{\mathbf{G}}_r \in \mathbb{R}^{1 \times L_2 \times \cdots \times L_N}$ and $\underline{\mathbf{D}}_r \in \mathbb{R}^{1 \times K_2 \times \cdots \times K_M}$ are core tensors. Note that the new tensor subspace for $\underline{\mathbf{X}}$ is spanned by $R$ tensor bases represented by Tucker model

$$\{\widetilde{\underline{\mathbf{P}}}_r\}_{r=1}^{R} = \underline{\mathbf{G}}_r \times_2 \mathbf{P}_r^{(1)} \times_3 \cdots \times_N \mathbf{P}_r^{(N-1)}, \tag{7}$$

while the new subspace for $\underline{\mathbf{Y}}$ is represented by Tucker model

$$\{\widetilde{\underline{\mathbf{Q}}}_r\}_{r=1}^{R} = \underline{\mathbf{D}}_r \times_2 \mathbf{Q}_r^{(1)} \times_3 \cdots \times_N \mathbf{Q}_r^{(M-1)}. \tag{8}$$

The rank-$(1, L_2, \ldots, L_N)$ decomposition in (6) is not unique, however, since MSVD generates both an all-orthogonal core [19] and column-wise orthogonal factors, these can be applied to obtain the unique components of the Tucker decomposition. This way, we ensure that $\underline{\mathbf{G}}_r$ and $\underline{\mathbf{D}}_r$ are all-orthogonal and $\mathbf{P}_r^{(n)}$, $\mathbf{Q}_r^{(m)}$ are column-wise orthogonal, i.e. $\mathbf{P}_r^{(n)T} \mathbf{P}_r^{(n)} = \mathbf{I} \in \mathbb{R}^{L_{n+1} \times L_{n+1}}$ and $\mathbf{Q}_r^{(m)T} \mathbf{Q}_r^{(m)} = \mathbf{I} \in \mathbb{R}^{K_{m+1} \times K_{m+1}}$.

By defining a latent matrix $\mathbf{T} = [\mathbf{t}_1, \ldots, \mathbf{t}_R] \in \mathbb{R}^{I_1 \times R}$, mode-$n$ loading matrix $\overline{\mathbf{P}}^{(n)} = [\mathbf{P}_1^{(n)}, \ldots, \mathbf{P}_R^{(n)}] \in \mathbb{R}^{I_{n+1} \times RL_{n+1}}$, mode-$m$ loading matrix $\overline{\mathbf{Q}}^{(m)} = [\mathbf{Q}_1^{(m)}, \ldots, \mathbf{Q}_R^{(m)}] \in \mathbb{R}^{J_{n+1} \times RK_{m+1}}$ and core tensor $\overline{\mathbf{G}} = \text{blockdiag}(\underline{\mathbf{G}}_1, \ldots, \underline{\mathbf{G}}_R) \in \mathbb{R}^{R \times RL_2 \times \cdots \times RL_N}$, $\overline{\mathbf{D}} = \text{blockdiag}(\underline{\mathbf{D}}_1, \ldots, \underline{\mathbf{D}}_R) \in \mathbb{R}^{R \times RK_2 \times \cdots \times RK_M}$, the HOPLS model in (6) can be rewritten as

$$\underline{\mathbf{X}} = \overline{\mathbf{G}} \times_1 \mathbf{T} \times_2 \overline{\mathbf{P}}^{(1)} \times \cdots \times_N \overline{\mathbf{P}}^{(N-1)} + \underline{\mathbf{E}},$$

$$\underline{\mathbf{Y}} = \overline{\mathbf{D}} \times_1 \mathbf{T} \times_2 \overline{\mathbf{Q}}^{(1)} \times_3 \cdots \times_M \overline{\mathbf{Q}}^{(M-1)} + \underline{\mathbf{F}}, \tag{9}$$

where $\underline{\mathbf{E}}$ and $\underline{\mathbf{F}}$ are residuals. The core tensors $\overline{\mathbf{G}}$ and $\overline{\mathbf{D}}$ have a special block-diagonal structure (see Fig. 1) whose elements indicate the level of interactions between the corresponding latent vectors and loading matrices.

Note that HOPLS simplifies into NPLS if we define $\forall n : \{L_n\} = 1$ and $\forall m : \{K_m\} = 1$. On the other hand, for $\forall n : \{L_n\} = \text{rank}_n(\underline{\mathbf{X}})$ and $\forall m : \{K_m\} = \text{rank}_m(\underline{\mathbf{Y}})^2$, HOPLS obtains the same solution as the standard PLS performed on a mode-1 matricization of $\underline{\mathbf{X}}$ and $\underline{\mathbf{Y}}$. This is obvious from a matricized form of (6), given by

$$\mathbf{X}_{(1)} \approx \mathbf{t}_r \mathbf{G}_{r(1)} \left( \mathbf{P}_r^{(N-1)} \otimes \cdots \otimes \mathbf{P}_r^{(1)} \right)^T, \tag{10}$$

where $\mathbf{G}_{r(1)} \left( \mathbf{P}_r^{(N-1)} \otimes \cdots \otimes \mathbf{P}_r^{(1)} \right)^T$ can approximate arbitrarily well the $\mathbf{p}_r^T$ in (4) computed from $\mathbf{X}_{(1)}$.

### 3.2 Objective function and algorithm

The optimization of subspace transformation yielding the common latent variables will be formulated as a problem of determining a set of loading matrices $\mathbf{P}_r^{(n)}, \mathbf{Q}_r^{(m)}, r = 1, 2, \ldots, R$ that maximize an objective function. Since the latent vectors can be optimized sequentially with the same

---

$^2 \text{rank}_n(\underline{\mathbf{X}}) = \text{rank}\left( \mathbf{X}_{(n)} \right).$

---

**Algorithm 1** The Higher-order Partial Least Squares (HOPLS) Algorithm

---

**Input:** $\underline{\mathbf{X}} \in \mathbb{R}^{I_1 \times \cdots \times I_N}, \underline{\mathbf{Y}} \in \mathbb{R}^{J_1 \times \cdots \times J_M}$ with $I_1 = J_1$
   Number of latent vectors is $R$ and number of loading vectors are $\{L_n\}_{n=2}^N$ and $\{K_m\}_{m=2}^M$.
**Output:** $\{\mathbf{P}_r^{(n)}\}; \{\mathbf{Q}_r^{(m)}\}; \{\underline{\mathbf{G}}_r\}; \{\underline{\mathbf{D}}_r\}; \mathbf{T}_r$
   $r = 1, \ldots, R; \ n = 1, \ldots, N-1; \ m = 1, \ldots, M-1.$
   **Initialization:** $\underline{\mathbf{E}}_1 = \underline{\mathbf{X}}, \quad \underline{\mathbf{F}}_1 = \underline{\mathbf{Y}}.$
   **for** $r = 1$ **to** $R$ **do**
     **if** $\|\underline{\mathbf{E}}_r\| > \epsilon$ **and** $\|\underline{\mathbf{F}}_r\| > \epsilon$ **then**
       $\underline{\mathbf{C}}_r \leftarrow <\underline{\mathbf{E}}_r, \underline{\mathbf{F}}_r>_{\{1,1\}};$
       Rank-$(L_2, \ldots, L_N, K_2, \ldots, K_M)$ decomposition of $\underline{\mathbf{C}}_r$ by HOOI [8] as
       $\underline{\mathbf{C}}_r \approx [\![\underline{\mathbf{H}}_r; \mathbf{P}_r^{(1)}, \ldots, \mathbf{P}_r^{(N-1)}, \mathbf{Q}_r^{(1)}, \ldots, \mathbf{Q}_r^{(M-1)}]\!];$
       $\mathbf{t}_r \leftarrow$ the first leading left singular vector by $\text{SVD} \left[\left(\underline{\mathbf{E}}_r \times_2 \mathbf{P}_r^{(1)T} \times_3 \cdots \times_N \mathbf{P}_r^{(N-1)T}\right)_{(1)}\right];$
       $\underline{\mathbf{G}}_r \leftarrow [\![\underline{\mathbf{E}}_r; \mathbf{t}_r^T, \mathbf{P}_r^{(1)T}, \ldots, \mathbf{P}_r^{(N-1)T}]\!];$
       $\underline{\mathbf{D}}_r \leftarrow [\![\underline{\mathbf{F}}_r; \mathbf{t}_r^T, \mathbf{Q}_r^{(1)T}, \ldots, \mathbf{Q}_r^{(M-1)T}]\!];$
       **Deflation**:
       $\underline{\mathbf{E}}_{r+1} \leftarrow \underline{\mathbf{E}}_r - [\![\underline{\mathbf{G}}_r; \mathbf{t}_r, \mathbf{P}_r^{(1)}, \ldots, \mathbf{P}_r^{(N-1)}]\!];$
       $\underline{\mathbf{F}}_{r+1} \leftarrow \underline{\mathbf{F}}_r - [\![\underline{\mathbf{D}}_r; \mathbf{t}_r, \mathbf{Q}_r^{(1)}, \ldots, \mathbf{Q}_r^{(M-1)}]\!];$
     **else**
       Break;
     **end if**
   **end for**
   Return all $\{\mathbf{P}_r^{(n)}\}; \{\mathbf{Q}_r^{(m)}\}; \{\underline{\mathbf{G}}_r\}; \{\underline{\mathbf{D}}_r\}; \mathbf{T}_r.$

---

criteria based on deflation[3], we shall simplify the problem to that of the first latent vector $\mathbf{t}_1$ and two groups of loading matrices $\mathbf{P}_1^{(n)}$ and $\mathbf{Q}_1^{(m)}$. To simplify the notation, $r$ is removed in the following equations. An objective function employed to determine the tensor bases, represented by $\mathbf{P}^{(n)}$ and $\mathbf{Q}^{(m)}$, can be defined as

$$\min_{\{\mathbf{P}^{(n)}, \mathbf{Q}^{(m)}\}} \left\| \underline{\mathbf{X}} - [\![\underline{\mathbf{G}}; \mathbf{t}, \mathbf{P}^{(1)}, \ldots, \mathbf{P}^{(N-1)}]\!]\right\|^2 + \left\| \underline{\mathbf{Y}} - [\![\underline{\mathbf{D}}; \mathbf{t}, \mathbf{Q}^{(1)}, \ldots, \mathbf{Q}^{(M-1)}]\!]\right\|^2$$

s. t. $\quad \{\mathbf{P}^{(n)T}\mathbf{P}^{(n)}\} = \mathbf{I}_{L_{n+1}}, \quad \{\mathbf{Q}^{(m)T}\mathbf{Q}^{(m)}\} = \mathbf{I}_{K_{m+1}},$          (11)

and yields the common latent vector $\mathbf{t}$ that best approximates $\underline{\mathbf{X}}$ and $\underline{\mathbf{Y}}$. The solution can be obtained by maximizing the norm of the core tensors $\underline{\mathbf{G}}$ and $\underline{\mathbf{D}}$ simultaneously. Since $\mathbf{t}^T \mathbf{t} = 1$, we have

$$\|\underline{\mathbf{G}} \times_1 \underline{\mathbf{D}}\|^2 = \left\| [\![<\underline{\mathbf{X}}, \underline{\mathbf{Y}}>_{\{1;1\}}; \mathbf{P}^{(1)}, \ldots, \mathbf{P}^{(N-1)}, \mathbf{Q}^{(1)}, \ldots \mathbf{Q}^{(M-1)}]\!]\right\|^2. \quad\quad (12)$$

We now define a mode-1 cross-covariance tensor $\underline{\mathbf{C}} = \text{COV}_{\{1;1\}}(\underline{\mathbf{X}}, \underline{\mathbf{Y}}) \in \mathbb{R}^{I_2 \times \cdots \times I_N \times J_2 \times \cdots \times J_M}$. Using the property of $\|\underline{\mathbf{G}} \times_1 \underline{\mathbf{D}}\|^2 \leq \|\underline{\mathbf{G}}\|^2 \|\underline{\mathbf{D}}\|^2$ and based on (11), (12), we have

$$\max_{\{\mathbf{P}^{(n)}, \mathbf{Q}^{(m)}\}} \left\| [\![\underline{\mathbf{C}}; \mathbf{P}^{(1)}, \ldots, \mathbf{P}^{(N-1)}, \mathbf{Q}^{(1)}, \ldots \mathbf{Q}^{(M-1)}]\!]\right\|^2$$

s. t. $\quad \mathbf{P}^{(n)T}\mathbf{P}^{(n)} = \mathbf{I}_{L_{n+1}}$ and $\mathbf{Q}^{(n)T}\mathbf{Q}^{(n)} = \mathbf{I}_{K_{m+1}},$          (13)

indicating that instead of decomposing $\underline{\mathbf{X}}$ directly, we may opt to find a rank-$(L_2, \ldots, L_N, K_2, \ldots, K_M)$ tensor decomposition of $\underline{\mathbf{C}}$. According to (11), for a given set of loading matrices $\{\mathbf{P}^{(n)}\}$, the latent vector $\mathbf{t}$ must explain variance of $\underline{\mathbf{X}}$ as much as possible, that is

$$\mathbf{t} = \arg\min_{\mathbf{t}} \left\| \underline{\mathbf{X}} - [\![\underline{\mathbf{G}}; \mathbf{t}, \mathbf{P}^{(1)}, \ldots, \mathbf{P}^{(N-1)}]\!]\right\|^2. \quad\quad (14)$$

The HOPLS algorithm is outlined in Algorithm 1.

## 3.3 Prediction

Predictions of the new observations are performed using the matricization form of data tensors $\underline{\mathbf{X}}$ and $\underline{\mathbf{Y}}$. More specifically, for any new observation $\underline{\mathbf{X}}^{new}$, we can predict the $\underline{\mathbf{Y}}^{new}$ as

$$\hat{\mathbf{T}}^{new} = \mathbf{X}_{(1)}^{new} \left( \overline{\mathbf{P}}^{(N-1)} \otimes \cdots \otimes \overline{\mathbf{P}}^{(1)} \right) \left( \overline{\mathbf{G}}_{(1)}^{T} \right)^{+}$$

$$\hat{\mathbf{Y}}_{(1)}^{new} = \hat{\mathbf{T}}^{new} \overline{\mathbf{D}}_{(1)} \left( \overline{\mathbf{Q}}^{(M-1)} \otimes \cdots \otimes \overline{\mathbf{Q}}^{(1)} \right)^{T}, \tag{15}$$

where $(\cdot)^{+}$ denotes the Moore-Penrose pseudoinverse operation.

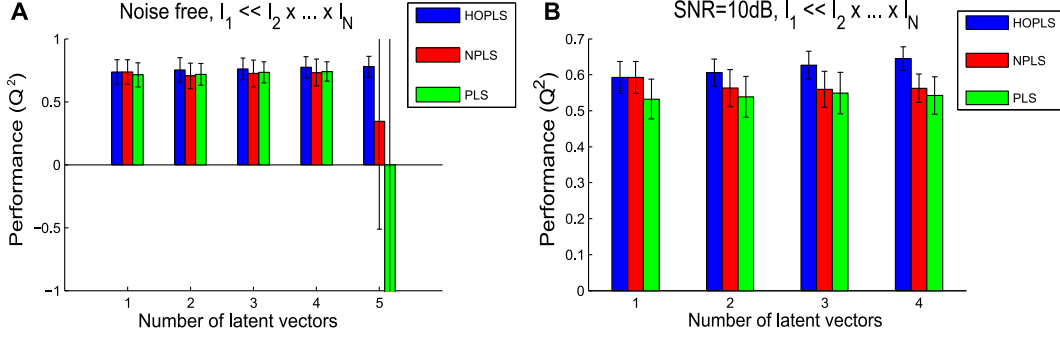

Figure 2: Performance comparison between HOPLS, NPLS and PLS, for a varying number of latent vectors under the conditions of noise free (A) and SNR=10dB (B).

## 4 Experimental results

We performs two case studies, one on synthetic data which illustrates the benefits of HOPLS, and the other on real-life electrophysiological data. To quantify the predictability the index $Q^2$ was defined as $Q^2 = 1 - \sum_{i=1}^{I}(y_i - \hat{y}_i)^2 / \sum_{i=1}^{I}(y_i - \bar{y})^2$, where $\hat{y}_i$ denotes the prediction of $y_i$ using a model created with the $i$th sample omitted.

### 4.1 Simulations on synthetic datasets

A simulation study on synthetic datasets was undertaken to evaluate the HOPLS regression method in terms of its predictive ability and effectiveness under different conditions related to small number of samples and noise levels. The HOPLS and NPLS were performed on tensor datasets whereas

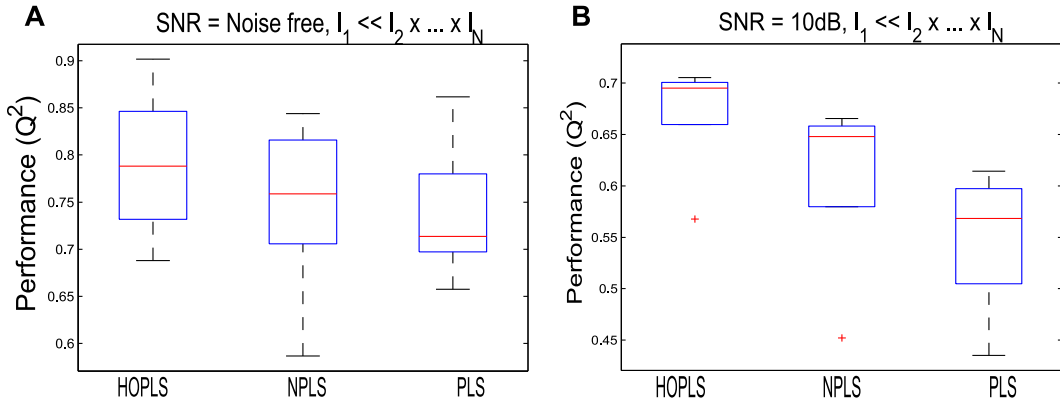

Figure 3: The optimal performance after choosing an appropriate number of latent vectors. (A) Noise free case. (B) For case with SNR=10dB.

PLS was performed on a mode-1 matricization of the corresponding datasets (i.e. $\mathbf{X}_{(1)}$ and $\mathbf{Y}_{(1)}$). The tensor $\underline{\mathbf{X}}$ was generated from a full-rank standard normal distribution and the tensor $\underline{\mathbf{Y}}$ as a linear combination of $\underline{\mathbf{X}}$. Noise was added to both independent and dependent datasets to evaluate performance at different noise levels. To reduce random fluctuations, the results were averaged over 50 simulation trials with datasets generated repeatedly according to the same criteria.

We considered a 3th-order tensor $\underline{\mathbf{X}}$ and a 3th-order tensor $\underline{\mathbf{Y}}$, for the case where the sample size was much smaller than the number of predictors, i.e., $I_1 << I_2 \times I_3$. Fig. 2 illustrates the predictive performances on the validation datasets for a varying number of latent vectors. Observe that when the number of latent vectors was equal to the number of samples, both PLS and NPLS had the tendency to be unstable, while HOPLS had no such problems. With an increasing number of latent vectors, HOPLS exhibited enhanced performance while the performance of NPLS and PLS deteriorated due to the noise introduced by excess latent vectors (see Fig. 2B). Fig. 3 illustrates the optimal prediction performances obtained by selecting an appropriate number of latent vectors. The HOPLS outperformed the NPLS and PLS at different noise levels and the superiority of HOPLS was more pronounced in the presence of noise, indicating its enhanced robustness to noise.

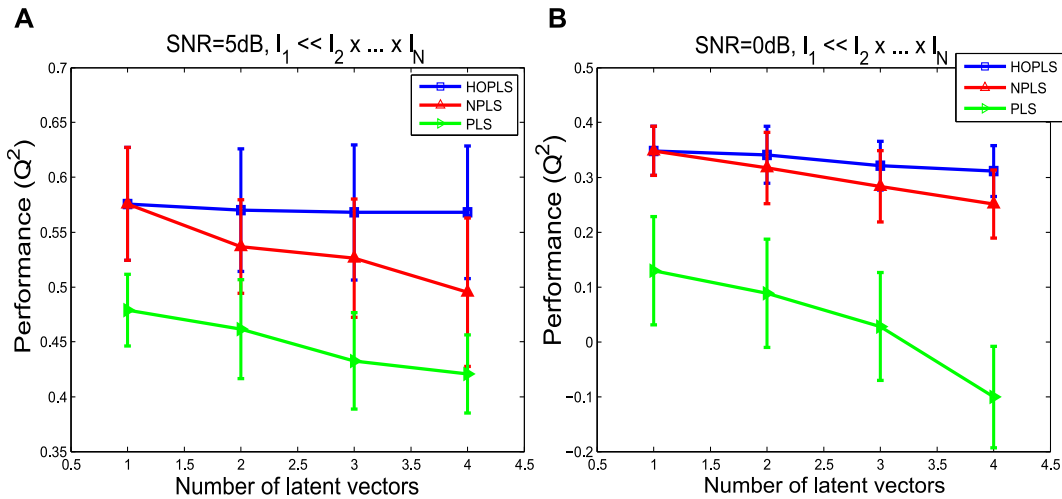

Figure 4: Stability of the performance of HOPLS, NPLS and PLS for a varying number of latent vectors, under the conditions of (A) SNR=5dB and (B) SNR=0dB.

Observe that PLS was sensitive to the number of latent vectors, indicating that the selection of latent vectors is a crucial issue for obtaining an optimal model. Finding the optimal number of latent vectors for unseen test data remains a challenging problem, implying that the stability of prediction performance for a varying number of latent vectors is essential for alleviating the sensitivity of the model. Fig. 4 illustrates the stable predictive performance of HOPLS for a varying number of latent vectors, this behavior was more pronounced for higher noise levels.

## 4.2 Decoding ECoG from EEG

In the last decade, considerable progress has been made in decoding the movement kinematics (e.g. trajectories or velocity) from neuronal signals recorded both invasively, such as spiking activity [20] and electrocorticogram (ECoG) [21, 22], and noninvasively- from scalp electroencephalography (EEG) [23]. To extract more information from brain activities, neuroimaging data fusion has also been investigated, whereby mutimodal brain activities were recorded continuously and synchronously. In contrast to the task of decoding the behavioral data from brain activity, in this study, our aim was to decode intracranial ECoG from scalp EEG. Assuming that both ECoG and EEG are related to the same brain sources, we set out to extract the common latent components between EEG and ECoG and examined whether ECoG can be decoded from the corresponding EEG by employing our proposed HOPLS method.

ECoG ($8\times8$ grid) and EEG (21 electrodes) were recorded simultaneously at a sample rate of 1024Hz from a human subject during relaxed state. After the preprocessing by spatial filter of common aver-

age reference (CAR), ECoG and EEG signals were transformed into a time-frequency representation and downsampled to 8 Hz by the continuous complex Morlet wavelet transformation with frequency range of 2-150Hz and 2-40Hz, respectively. To ease the computation burden, we employed a 4 second time window of EEG to predict the corresponding ECoG with the same window length. Thus, our objective was to decode the ECoG dataset comprised in a 4th-order tensor $\underline{\mathbf{Y}}$ (trial $\times$ channel $\times$ frequency $\times$ time) from an EEG dataset contained in a 4th-order tensor $\underline{\mathbf{X}}$ (trial $\times$ channel $\times$ frequency $\times$ time).

According to the HOPLS model, the common latent vectors in $\mathbf{T}$ can be regarded as brain source components that establish a bridge between EEG and ECoG, while the loading tensors $\widetilde{\underline{\mathbf{P}}}_r$ and $\widetilde{\underline{\mathbf{Q}}}_r, r = 1, \ldots, R$ can be regarded as a set of tensor bases, as shown in Fig. 5(A). These bases are computed from the training dataset and explain the relationship of spatio-temporal frequency patterns between EEG and ECoG. The decoding model was calibrated from 30 second datasets and was applied to predict the subsequent 30 second datasets. The quality of prediction was evaluated by the values of total correlation coefficients between the predicted and actual time-frequency representation of ECoG, denoted by $r_{\mathrm{vec}(\hat{\underline{\mathbf{Y}}}), \mathrm{vec}(\underline{\mathbf{Y}})}$.

Fig. 5(B) illustrates the prediction performance by using a different number of latent vectors, ranging from 1 to 8 and compared with the standard PLS performed on a mode-1 matricization of tensors $\underline{\mathbf{X}}$ and $\underline{\mathbf{Y}}$. The optimal number of latent vectors for HOPLS and PLS were 4 and 1, respectively. Conforming with analysis, HOPLS was more stable for a varying number of latent vectors and outperformed the standard PLS in terms of its predictive ability.

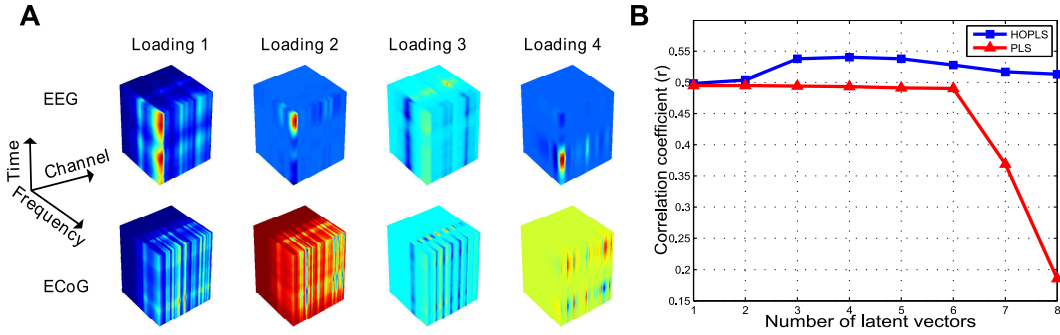

Figure 5: (A) The basis of the tensor subspace computed from the spatial, temporal, and spectral representation of EEG and ECoG. (B) The correlation coefficient $r$ between predicted and actual spatio-temporal-frequency representation of ECoG signals for a varying number of latent vectors.

## 5   Conclusion

We have introduced the Higher-order Partial Least Squares (HOPLS) framework for tensor subspace regression, whereby data samples are represented in a tensor form, thus providing an natural generalization of the existing Partial Least Squares (PLS) and $N$-way PLS (NPLS) approaches. Compared to the standard PLS, our proposed method has been shown to be more flexible and robust, especially for small sample size cases. Simulation results have demonstrated the superiority and effectiveness of HOPLS over the existing algorithms for different noise levels. A challenging application of decoding intracranial electrocorticogram (ECoG) from a simultaneously recorded scalp electroencephalography (EEG) (both from human brain) has been studied and the results have demonstrated the large potential of HOPLS for multi-way correlated datasets.

**Acknowledgments**

The work was supported in part by the national natural science foundation of China under grant number 90920014 and NSFC international cooperation program under grant number 61111140019.

## Footnotes

[1]The first mode is usually associated with the sample mode or time mode, and for each sample, we have an independent data represented by an $(N-1)$th-order tensor and a dependent data represented by an $(M-1)$th-order tensor.

[3]As in the NPLS case, this deflation does not reduce the rank of the residuals.

# References

[1] L. Wolf, H. Jhuang, and T. Hazan. Modeling appearances with low-rank SVM. In *IEEE Conference on Computer Vision and Pattern Recognition*, pages 1–6. IEEE, 2007.

[2] Hamed Pirsiavash, Deva Ramanan, and Charless Fowlkes. Bilinear classifiers for visual recognition. In Y. Bengio, D. Schuurmans, J. Lafferty, C. K. I. Williams, and A. Culotta, editors, *Advances in Neural Information Processing Systems 22*, pages 1482–1490. 2009.

[3] H. Lu, K.N. Plataniotis, and A.N. Venetsanopoulos. MPCA: Multilinear principal component analysis of tensor objects. *IEEE Transactions on Neural Networks*, 19(1):18–39, 2008.

[4] S. Yan, D. Xu, Q. Yang, L. Zhang, X. Tang, and H.J. Zhang. Multilinear discriminant analysis for face recognition. *IEEE Transactions on Image Processing*, 16(1):212–220, 2007.

[5] D. Tao, X. Li, X. Wu, and S.J. Maybank. General tensor discriminant analysis and Gabor features for gait recognition. *IEEE Transactions on Pattern Analysis and Machine Intelligence*, 29(10):1700–1715, 2007.

[6] A.K. Smilde and H.A.L. Kiers. Multiway covariates regression models. *Journal of Chemometrics*, 13(1):31–48, 1999.

[7] X. He, D. Cai, and P. Niyogi. Tensor subspace analysis. *Advances in Neural Information Processing Systems*, 18:499, 2006.

[8] T.G. Kolda and B.W. Bader. Tensor decompositions and applications. *SIAM Review*, 51(3):455–500, 2009.

[9] A. Cichocki, R. Zdunek, A. H. Phan, and S. I. Amari. *Nonnegative Matrix and Tensor Factorizations*. John Wiley & Sons, 2009.

[10] E. Acar, D.M. Dunlavy, T.G. Kolda, and M. Mørup. Scalable tensor factorizations for incomplete data. *Chemometrics and Intelligent Laboratory Systems*, 2010.

[11] R. Bro, R.A. Harshman, N.D. Sidiropoulos, and M.E. Lundy. Modeling multi-way data with linearly dependent loadings. *Journal of Chemometrics*, 23(7-8):324–340, 2009.

[12] S. Wold, M. Sjostroma, and L. Erikssonb. PLS-regression: A basic tool of chemometrics. *Chemometrics and Intelligent Laboratory Systems*, 58:109–130, 2001.

[13] H. Wold. Soft modeling by latent variables: The nonlinear iterative partial least squares approach. *Perspectives in probability and statistics, papers in honour of MS Bartlett*, pages 520–540, 1975.

[14] A. Krishnan, L.J. Williams, A.R. McIntosh, and H. Abdi. Partial least squares (PLS) methods for neuroimaging: A tutorial and review. *NeuroImage*, 56(2):455 – 475, 2011.

[15] H. Abdi. Partial least squares regression and projection on latent structure regression (PLS Regression). *Wiley Interdisciplinary Reviews: Computational Statistics*, 2(1):97–106, 2010.

[16] R. Rosipal and N. Krämer. Overview and recent advances in partial least squares. In *Subspace, Latent Structure and Feature Selection*, volume 3940 of *Lecture Notes in Computer Science*, pages 34–51. Springer, 2006.

[17] R. Bro. Multiway calibration. Multilinear PLS. *Journal of Chemometrics*, 10(1):47–61, 1996.

[18] L. De Lathauwer. Decompositions of a higher-order tensor in block terms - Part II: Definitions and uniqueness. *SIAM J. Matrix Anal. Appl*, 30(3):1033–1066, 2008.

[19] L. De Lathauwer, B. De Moor, and J. Vandewalle. A multilinear singular value decomposition. *SIAM Journal on Matrix Analysis and Applications*, 21(4):1253–1278, 2000.

[20] M. Velliste, S. Perel, M.C. Spalding, A.S. Whitford, and A.B. Schwartz. Cortical control of a prosthetic arm for self-feeding. *Nature*, 453(7198):1098–1101, 2008.

[21] Z.C. Chao, Y. Nagasaka, and N. Fujii. Long-term asynchronous decoding of arm motion using electrocorticographic signals in monkeys. *Frontiers in Neuroengineering*, 3(3), 2010.

[22] T. Pistohl, T. Ball, A. Schulze-Bonhage, A. Aertsen, and C. Mehring. Prediction of arm movement trajectories from ECoG-recordings in humans. *Journal of Neuroscience Methods*, 167(1):105–114, 2008.

[23] T.J. Bradberry, R.J. Gentili, and J.L. Contreras-Vidal. Reconstructing three-dimensional hand movements from noninvasive electroencephalographic signals. *The Journal of Neuroscience*, 30(9):3432, 2010.

